# Lower Boundaries of Motoneuron Desynchronization via Renshaw Interneurons

**Mitchell Gil Maltenfort***
Dept. of Biomedical Engineering
Northwestern University
Evanston, IL 60201

**Robert E. Druzinsky**
Dept. of Physiology
Northwestern University
Chicago, IL 60611

**C. J. Heckman**
V. A. Research Service
Lakeside Hospital
and Dept. of Physiology
Northwestern University
Chicago, IL 60611

**W. Zev Rymer**
Dept. of Physiology
and Biomedical Engineering
Northwestern University
Chicago, IL 60611

## Abstract

Using a quasi-realistic model of the feedback inhibition of motoneurons (MNs) by Renshaw cells, we show that weak inhibition is sufficient to maximally desynchronize MNs, with negligible effects on total MN activity. MN synchrony can produce a 20 - 30 Hz peak in the force power spectrum, which may cause instability in feedback loops.

## 1    INTRODUCTION

The structure of the recurrent inhibitory connections from Renshaw cells (RCs) onto motoneurons (MNs) (Figure 1) suggests that the RC forms a simple negative feedback

loop. Past theoretical work has examined possible roles of this feedback in smoothing or gain regulation of motor output (e.g., Bullock and Contreras-Vidal, 1991; Graham and Redman, 1993), but has assumed relatively strong inhibitory effects from the RC. Experimental observations (Granit et al.,1961) show that maximal RC activity can only reduce MN firing rates by a few impulses per second, although this weak inhibition is sufficient to affect the timing of MN firings, reducing the probability that any two MNs will fire simultaneously (Adam et al., 1978; Windhorst et al., 1978). In this study, simulations were used to examine the impact of RC inhibition on MN firing synchrony and to predict the effects of such synchrony on force output.

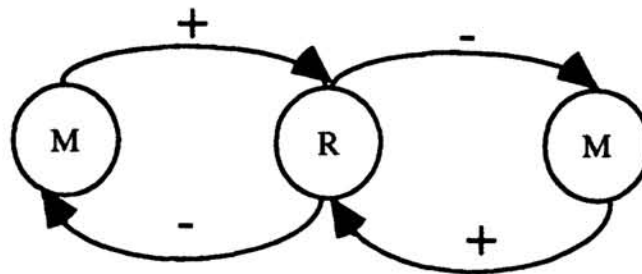

Figure 1: Simplified Schematic of Recurrent Inhibition

## 2     CONSTRUCTION OF THE MODEL

### 2.1     MODELING OF INDIVIDUAL NEURONS

The integrate-and-fire neuron model of MacGregor (1987) adequately mimics specific firing patterns. Coupled first-order differential equations govern membrane potential and afterhyperpolarization (AHP) based on injected current and synaptic inputs. A spike is fired when the membrane potential crosses a threshold. The model was modified to include a membrane resistance in order to model MNs of varying current thresholds.

Membrane resistance and time constants of model MNs were set to match published data (Gustaffson and Pinter, 1984). The parameters governing AHPs were adjusted to agree with observations from single action potentials and steady-state current-rate plots (Heckman and Binder, 1991). Realistic firing behavior could be generated for MNs with current thresholds of 4 - 40 nA.

Although there are no direct measurements of RC membrane properties available, appropriate parameters were estimated by extrapolation from the MN parameter set. The simulated RC has a 30 ms AHP and a current-rate plot matching that reported by Hultborn and Pierrot-Deseilligny (1979). Spontaneous firing of 8 pps is produced in the model by setting the RC firing threshold to 0.01 mV below resting potential; in vivo

this firing is likely due to descending inputs (Hamm et al., 1987a), but there is no quantitative description of such inputs. The RCs are assumed to be homgeneous.

## 2.2    CONNECTIVITY OF THE POOL

Simulated neurons were arranged along a 16 by 16 grid. The network consists of 256 MNs and 64 RCs, with the RCs ordered on even-numbered rows and columns; as a result, the MN - RC connections are inhomogeneous along the pool. For each trial, MN pools were randomly generated following the distribution of MN current thresholds for a model of the cat medial gastrocnemius motor pool (Heckman and Binder, 1991).

Communication between neurons is mediated by synaptic conductances which open when a presynaptic cell fires, then decay exponentially. MN excitation of RCs was set to produce RC firing rates $\leq$ 190 pps (Cleveland et al., 1981) which linearly increase with MN activity (Cleveland and Ross, 1977). MN activation of RCs scales inversely with MN current threshold (Hultborn et al., 1988).

Connectivity is based on observations that synapses from RCs to MNs have a longer spatial range than the reverse (reviewed in Windhorst, 1990). The IPSPs produced by single MN firings are 4 - 6 times larger than those produced by single RC firings (Hamm et al., 1987b; van Kuelen, 1981). In the model, each MN excites RCs within one column or row of itself, and each RC inhibits MNs up to two rows or columns away; thus, each MN excites 1 - 4 RCs (mean 2.25) and receives feedback from 4 - 9 RCs (mean 6.25).

## 2.3    ACTIVATION OF THE POOL

The MNs are activated by applied step currents. Although this is not realistic, it is computationally efficient. An option in the simulation program allows for the addition of bandlimited noise to the activation current, to simulate a synchronizing common synaptic input. This signal has an rms value of 3% of the mean applied current and is low-pass filtered with a cutoff of 30 Hz. This allows us look at the effects due purely to RC activity and to establish which effects persist when the MN pool is being actively synchronized.

# 3    EFFECTS OF RC STRENGTH ON MN SYNCHRONY

## 3.1    DEFINITION OF SYNCHRONY COEFFICIENT

Consider the total number of spikes fired by the MN pool as a time series. During synchronous firing, the MN spikes will clump together and the time series will have regions of very many or very few MN spikes. When the MNs are desynchronized, the range of spike counts in each time bin will contract towards the mean. It follows that a simple measure of MN synchrony is the the coefficient of variation (c.v. = s.d. / mean) of the time series formed by the summed MN activity. Figure 2 shows typical MN pool firing before and after RC feedback inhibition is added; the changes in "clumping" described above are quite visible in the two plots.

## 3.2   "PLATEAU" OF DESYNCHRONIZATION

The magnitude of the synaptic conductance from RCs onto MNs was changed from zero to twice physiological in order to compare the effects of 'weak' and 'strong' recurrent inhibition.  At activation levels sufficient to recruit at least 70% of MNs in the pool (mean firing rate $\geq$ 15 pps), a surprising plateau effect was seen.  The synchrony coefficient fell off with RC synaptic conductance until the physiological level was reached, and then no further desynchronization was seen.  The effect persisted when synchronizing noise was added (Figure 3).  At activation levels sufficient to show this plateau, this "corner" inhibition level was always the same.

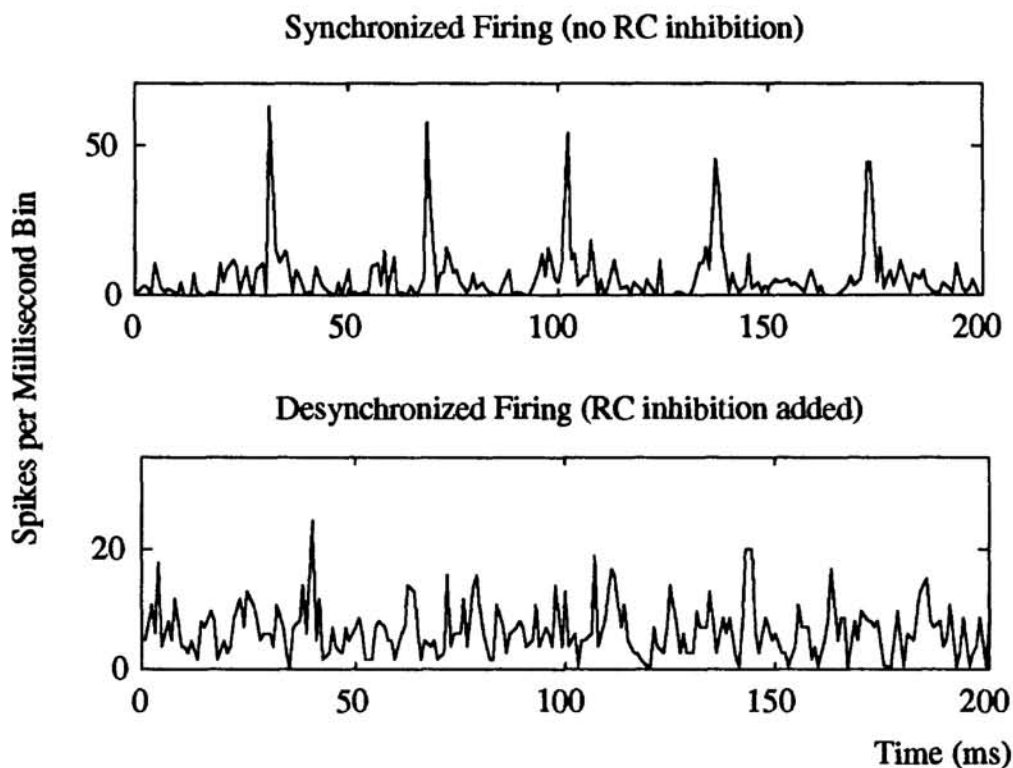

Figure 2: Comparison of Synchronous and Asynchronous MN Firing

At this corner level, the decrease in  mean MN firing rate was $\leq$ 1 pps and not statistically significant.  There was also no discernible change in the percentage of the MN pool active. The c.v. of the interspike interval of single MN firings during constant activation is $\leq$ 2.5 % even with RCs active - this implies that the RC system finds an optimal arrangement of the MN firings and then performs few if any further shifts. When synchronizing noise is added, the RC effect on the interspike interval is swamped by the effect of the synchronizing random input.

Figure 4 shows the effect of increasing MN activation on the synchrony coefficients before and after RC inhibition is added.  The change is statistically significant at all levels, but is only large at higher levels as discussed above.  As activation of the MNs

increases, the "before" level of synchrony increases while the "after" level seems to move asymptotically towards a minimum level of about 0.35. This minimum level of MN synchrony, as well as the dependence of the effect on the activation level of the pool, suggests that a certain amount of synchrony becomes inevitable as more MNs are activated and fire at higher rates.

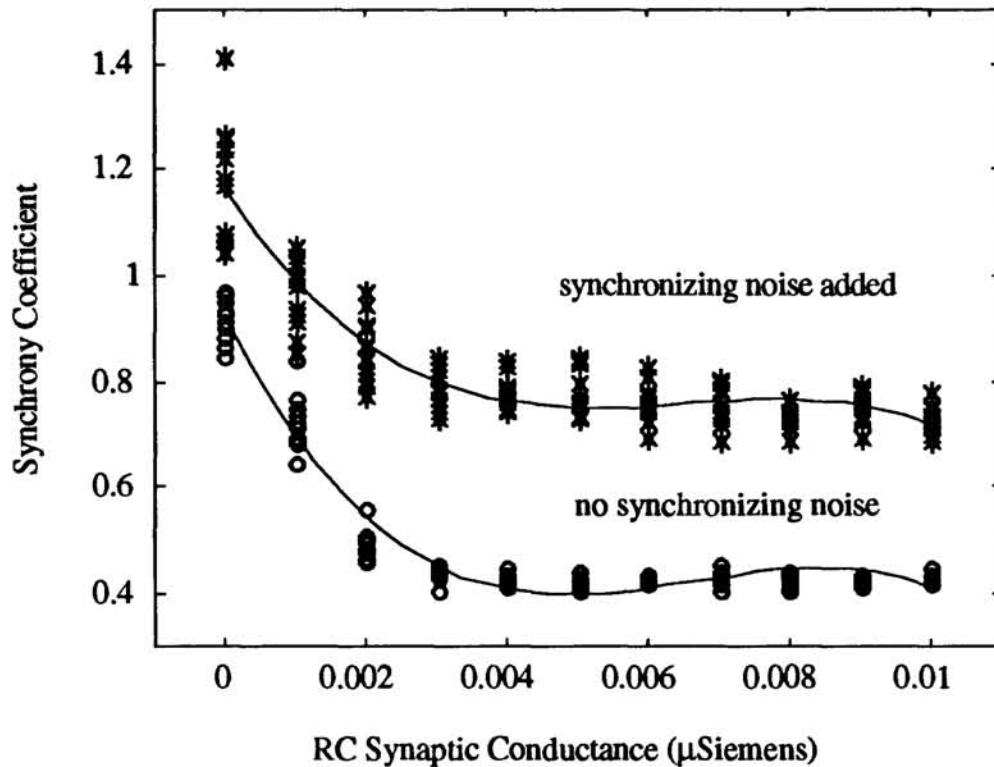

Figure 3: MN Firing Synchrony vs. RC Strength

## 4    EFFECTS OF MN SYNCHRONY ON MUSCLE FORCE

### 4.1    MODELING OF FORCE OUTPUT

Single twitches of motor units are modeled with a second-order model, $f(t) = \frac{Ft}{\tau}e^{-t/\tau}$, where the amplitude F and time constant $\tau$ are matched to MN current threshold according to the model of Heckman and Binder (1991). A rate-based gain factor adapted from Fuglevand (1989) produces fused tetanus at high firing rates. The tenfold difference in current thresholds maps to a fifty-fold difference in twitch forces. Twitch time constants range 30-90 ms.

## 4.2    EFFECTS OF RECURRENT INHIBITION ON FORCE

The force model sharply low-pass filters the neural input signal ($\leq$ 5 Hz). As a result, the c.v. of the force output is much lower than that of the associated MN input ($\leq$ 0.01). Although the plot of force c.v. vs. RC strength during constant activation follows the curve in Figure 3, adding synchronizing noise removes any correlation between force .c.v. and magnitude of recurrent inhibition. The effect of recurrent inhibition on mean force is similar to that on the mean firing rate: small ($\leq$5 % decrease) and generally not statistically significant.

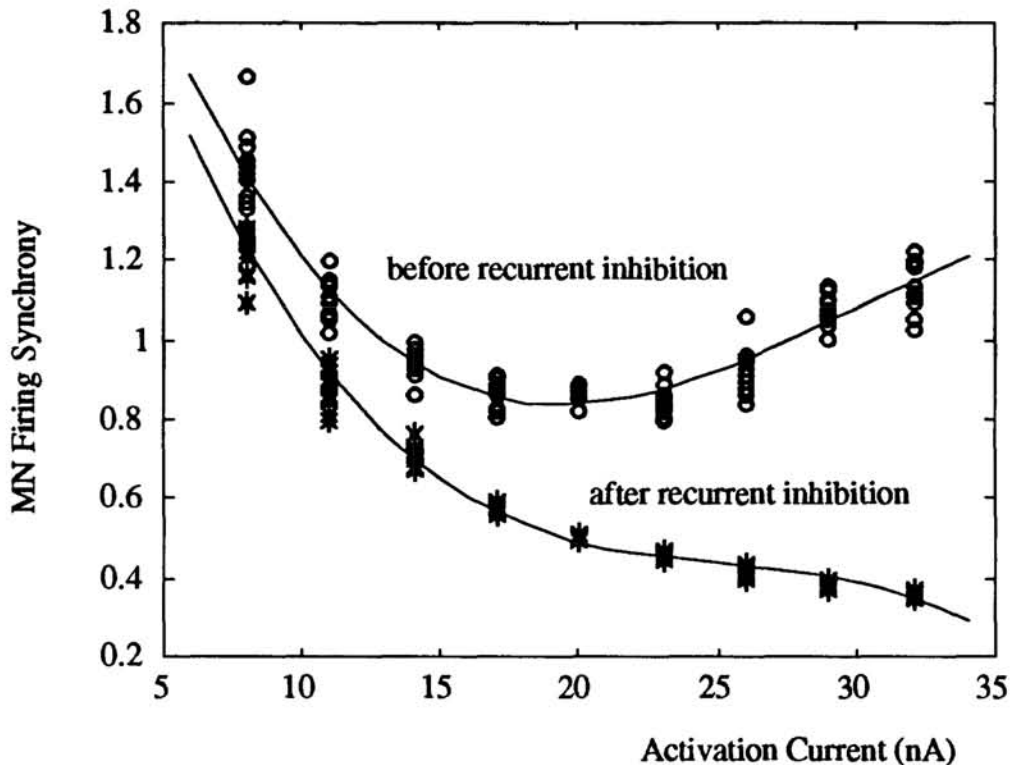

Figure 4: Effects of MN Activation on Synchrony Before and After Recurrent Inhibition

When the change in synchrony due to RCs is large, a peak appears in the force power spectrum in the range 20 - 30 Hz. This peak is reduced by RCs even when the MN pool is being actively synchronized (Figure 5). Peaks in the force spectrum match peaks in the spectrum of pooled MN activity, suggesting the effect is due to synchronous MN firing.

Although the magnitude of this peak is small ($\leq$ 0.5% of mean force), its relatively high frequency suggests that in derivative feedback - where spectral components are multiplied by $2\pi$ times their frequency - its impact could be substantial. The feedback loop which measures muscle stretch contains such a derivative component (Houk and Rymer, 1981).

## 5    DISCUSSION

The preceding shows that the ostensibly weak recurrent inhibition is sufficient to sharply reduce the maximum number of synchronous firings of a neuron population, while having a negligible effect on the total population activity. This has a broad implication for neural networks in that it suggests the existence of a "switching mechanism" which forces the peaks in the output of an ensemble of neurons to remain below a threshold level without significantly suppressing the total ensemble activity.

One possible role for such a mechanism would be in the accommodation to a step or ramp increase in a stimulus. The initial increase synchronizes the neural signal from the receptor, which is then desynchronized by the recurrent inhibition. The synchronized firing phase would be sufficient to excite a target neuron past its firing threshold, but after that, the desynchronized neural signal would remain well below the target's threshold.

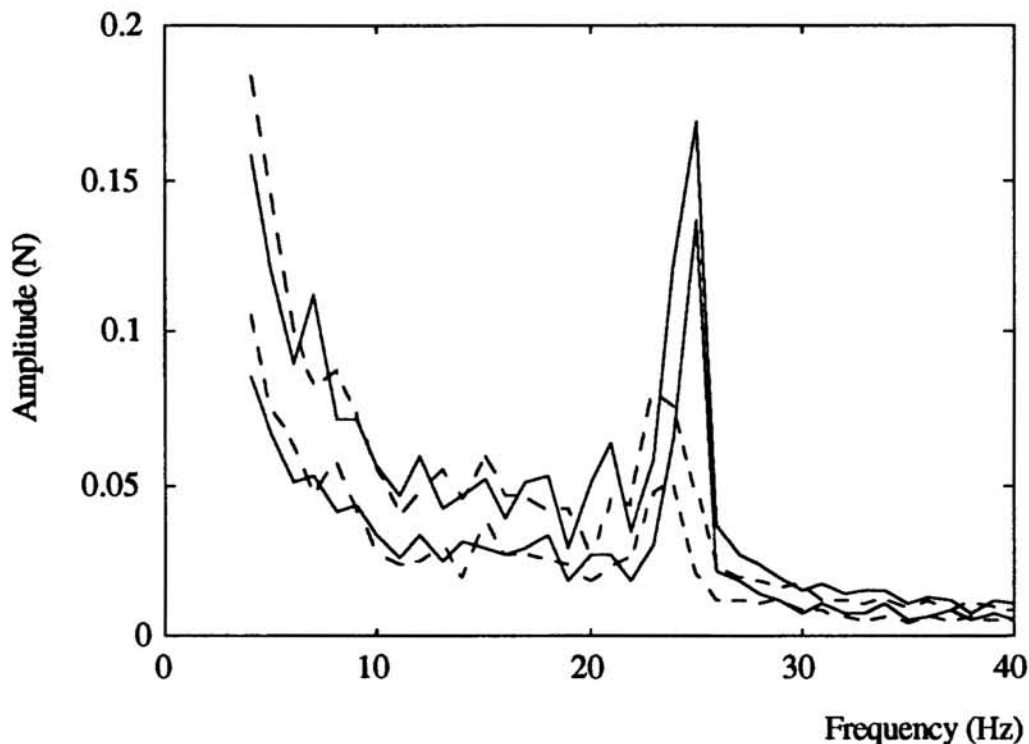

Figure 5: Recurrent Inhibition Reduces Spectral Peak. 95% confidence limit of means plotted, solid lines before recurrent inhibition and dashed lines after.

### Acknowledgments

The authors are indebted to Dr. Tom Buchanan for use of his IBM RS/6000 workstation. This work was supported by NIH grants NS28076-02 and NS30295-01.

## Footnotes

* send mail to: Mitchell G. Maltenfort, SMPP room 1406, Rehabilitation Insitute of Chicago, 345 East Superior Street, Chicago, IL 60611. Email address is mgm@nwu.edu

## References

Adam D, Windhorst U, Inbar GF: The effects of recurrent inhibition on the cross-correlated firing patterns of motoneurons (and their relation to signal transmission in the spinal cord-muscle channel). *Biol. Cybern.*, 29: 229-235, 1978.

Bullock D, Contreras-Vidal J: How spinal neural networks reduce discrepancies between motor intention and motor realization. Tech.Report CAS/CNS-91-023, Boston U., 1991.

Cleveland S, Kuschmierz A, Ross H-G: Static input-output relations in the spinal recurrent inhibitory pathway. *Biol. Cybern.*, 40: 223-231, 1981.

Cleveland S, Ross H-G: Dynamic properties of Renshaw cells: Frequency response characteristics. *Biol. Cybern.*, 27: 175-184, 1977.

Fuglevand AJ: A motor unit pool model: relationship of neural control properties to isometric muscle tension and the electromyogram. Ph.D. Thesis, U. of Waterloo, 1989.

Graham BP, Redman SJ: Dynamic behaviour of a model of the muscle stretch reflex. *Neural Networks*, 6: 947-962, 1993.

Granit R, Haase J, Rutledge LT: Recurrent inhibition in relation to frequency of firing and limitation of discharge rate of extensor motoneurons. *J. Physiol.*, 158: 461-475, 1961.

Gustaffson B, Pinter MJ: An investigation of threshold properties among cat spinal α-motoneurons. *J. Physiol.*, 357: 453-483, 1984.

Hamm TM, Sasaki S, Stuart DG, Windhorst U, Yuan C-S: Distribution of single-axon recurrent inhibitory post-synaptic potentials in the cat. *J. Physiol.*, 388: 631-651,1987a.

Hamm TM, Sasaki S, Stuart DG, Windhorst U, Yuan C-S: The measurement of single motor-axon recurrent inhibitory post-synaptic potentials in a single spinal motor nucleus in the cat. *J. Physiol.*, 388: 653-664, 1987b .

Heckman CJ, Binder MD: Computer simulation of the steady-state input-output function of the cat medial gastrocnemius motoneuron pool. *J. Neurophysiol.*, 65: 952-967, 1991.

Houk JC, Rymer WZ: Chapter 8: Neural control of muscle length and tension. In *Handbook of Physiology: the Nervous System II pt. 1*, ed.VB Brooks. Am. Physiol. Soc., Bethesda, MD, 1981.

Hultborn H, Peirrot-Deseillgny E: Input-output relations in the pathway of recurrent inhibition to motoneurons in the cat. *J. Physiol.*, 297: 267-287, 1979.

MacGregor RJ: *Neural and Brain Modeling.* Academic Press, San Diego, 1987.

Van Kuelen LCM: Autogenetic recurrent inhibition of individual spinal motoneurons of the cat. *Neurosci. Lett.*, 21: 297-300, 1981.

Windhorst U: Activation of Renshaw cells. *Prog. in Neurobiology*, 35: 135-179, 1990.

Windhorst U, Adam D, Inbar GF: The effects of recurrent inhibitory feedback in shaping discharge patterns of motoneurones excited by phasic muscle stretches. *Biol. Cybern.*, 29: 221-227, 1978.